# Subsequence Kernels for Relation Extraction

**Razvan C. Bunescu**
Department of Computer Sciences
University of Texas at Austin
1 University Station C0500
Austin, TX 78712
razvan@cs.utexas.edu

**Raymond J. Mooney**
Department of Computer Sciences
University of Texas at Austin
1 University Station C0500
Austin, TX 78712
mooney@cs.utexas.edu

## Abstract

We present a new kernel method for extracting semantic relations between entities in natural language text, based on a generalization of subsequence kernels. This kernel uses three types of subsequence patterns that are typically employed in natural language to assert relationships between two entities. Experiments on extracting protein interactions from biomedical corpora and top-level relations from newspaper corpora demonstrate the advantages of this approach.

## 1 Introduction

Information Extraction (IE) is an important task in natural language processing, with many practical applications. It involves the analysis of text documents, with the aim of identifying particular types of entities and relations among them. Reliably extracting relations between entities in natural-language documents is still a difficult, unsolved problem. Its inherent difficulty is compounded by the emergence of new application domains, with new types of narrative that challenge systems developed for other, well-studied domains. Traditionally, IE systems have been trained to recognize names of people, organizations, locations and relations between them (MUC [1], ACE [2]). For example, in the sentence "*protesters seized several pumping stations*", the task is to identify a LOCATED AT relationship between *protesters* (a PERSON entity) and *stations* (a LOCATION entity). Recently, substantial resources have been allocated for automatically extracting information from biomedical corpora, and consequently much effort is currently spent on automatically identifying biologically relevant entities, as well as on extracting useful biological relationships such as protein interactions or subcellular localizations. For example, the sentence "*TR6 specifically binds Fas ligand*", asserts an interaction relationship between the two proteins *TR6* and *Fas ligand*. As in the case of the more traditional applications of IE, systems based on manually developed extraction rules [3, 4] were soon superseded by information extractors learned through training on supervised corpora [5, 6]. One challenge posed by the biological domain is that current systems for doing part-of-speech (POS) tagging or parsing do not perform as well on the biomedical narrative as on the newspaper corpora on which they were originally trained. Consequently, IE systems developed for biological corpora need to be robust to POS or parsing errors, or to give reasonable performance using shallower but more reliable information, such as chunking instead of parsing.

Motivated by the task of extracting protein-protein interactions from biomedical corpora, we present a generalization of the subsequence kernel from [7] that works with sequences containing combinations of words and word classes. This generalized kernel is further tailored for the task of relation extraction. Experimental results show that the new relation

kernel outperforms two previous rule-based methods for interaction extraction. With a small modification, the same kernel is used for extracting top-level relations from ACE corpora, providing better results than a recent approach based on dependency tree kernels.

## 2  Background

One of the first approaches to extracting protein interactions is that of Blaschke *et al.*, described in [3, 4]. Their system is based on a set of manually developed rules, where each rule (or frame) is a sequence of words (or POS tags) and two protein-name tokens. Between every two adjacent words is a number indicating the maximum number of intervening words allowed when matching the rule to a sentence. An example rule is "*interaction of (3) <P> (3) with (3) <P>*", where '<P>' is used to denote a protein name. A sentence matches the rule if and only if it satisfies the word constraints in the given order and respects the respective word gaps.

In [6] the authors described a new method ELCS (Extraction using Longest Common Subsequences) that automatically learns such rules. ELCS' rule representation is similar to that in [3, 4], except that it currently does not use POS tags, but allows disjunctions of words. An example rule learned by this system is "- *(7) interaction (0) [between | of] (5) <P> (9) <P> (17)*.". Words in square brackets separated by '|' indicate disjunctive lexical constraints, i.e. one of the given words must match the sentence at that position. The numbers in parentheses between adjacent constraints indicate the maximum number of unconstrained words allowed between the two.

## 3  Extraction using a Relation Kernel

Both Blaschke and ELCS do interaction extraction based on a limited set of matching rules, where a rule is simply a sparse (gappy) subsequence of words or POS tags anchored on the two protein-name tokens. Therefore, the two methods share a common limitation: either through manual selection (Blaschke), or as a result of the greedy learning procedure (ELCS), they end up using only a subset of all possible anchored sparse subsequences. Ideally, we would want to use all such anchored sparse subsequences as features, with weights reflecting their relative accuracy. However explicitly creating for each sentence a vector with a position for each such feature is infeasible, due to the high dimensionality of the feature space. Here we can exploit dual learning algorithms that process examples only via computing their dot-products, such as the Support Vector Machines (SVMs) [8]. Computing the dot-product between two such vectors amounts to calculating the number of common anchored subsequences between the two sentences. This can be done very efficiently by modifying the dynamic programming algorithm used in the string kernel from [7] to account only for common sparse subsequences constrained to contain the two protein-name tokens. We further prune down the feature space by utilizing the following property of natural language statements: when a sentence asserts a relationship between two entity mentions, it generally does this using one of the following three patterns:

- **[FB] F**ore–**B**etween: words before and between the two entity mentions are simultaneously used to express the relationship. Examples: 'interaction of $\langle P_1 \rangle$ with $\langle P_2 \rangle$', 'activation of $\langle P_1 \rangle$ by $\langle P_2 \rangle$'.

- **[B] B**etween: only words between the two entities are essential for asserting the relationship. Examples: '$\langle P_1 \rangle$ interacts with $\langle P_2 \rangle$', '$\langle P_1 \rangle$ is activated by $\langle P_2 \rangle$'.

- **[BA] B**etween–**A**fter: words between and after the two entity mentions are simultaneously used to express the relationship. Examples: '$\langle P_1 \rangle - \langle P_2 \rangle$ complex', '$\langle P_1 \rangle$ and $\langle P_2 \rangle$ interact'.

Another observation is that all these patterns use at most 4 words to express the relationship (not counting the two entity names). Consequently, when computing the relation kernel, we restrict the counting of common anchored subsequences only to those having one of the three types described above, with a maximum word-length of 4. This type of feature

selection leads not only to a faster kernel computation, but also to less overfitting, which results in increased accuracy (see Section 5 for comparative experiments).

The patterns enumerated above are completely lexicalized and consequently their performance is limited by data sparsity. This can be alleviated by categorizing words into classes with varying degrees of generality, and then allowing patterns to use both words and their classes. Examples of word classes are POS tags and generalizations over POS tags such as Noun, Active Verb or Passive Verb. The entity type can also be used, if the word is part of a known named entity, as well as the type of the chunk containing the word, when chunking information is available. Content words such as nouns and verbs can also be related to their synsets via WordNet. Patterns then will consist of sparse subsequences of words, POS tags, general POS (GPOS) tags, entity and chunk types, or WordNet synsets. For example, 'Noun of $\langle P_1 \rangle$ by $\langle P_2 \rangle$' is an FB pattern based on words and general POS tags.

## 4  Subsequence Kernels for Relation Extraction

We are going to show how to compute the relation kernel described in the previous section in two steps. First, in Section 4.1 we present a generalization of the subsequence kernel from [7]. This new kernel works with patterns construed as mixtures of words and word classes. Based on this generalized subsequence kernel, in Section 4.2 we formally define and show the efficient computation of the relation kernel used in our experiments.

### 4.1  A Generalized Subsequence Kernel

Let $\Sigma_1, \Sigma_2, ..., \Sigma_k$ be some disjoint feature spaces. Following the example in Section 3, $\Sigma_1$ could be the set of words, $\Sigma_2$ the set of POS tags, etc. Let $\Sigma_\times = \Sigma_1 \times \Sigma_2 \times ... \times \Sigma_k$ be the set of all possible feature vectors, where a feature vector would be associated with each position in a sentence. Given two feature vectors $x, y \in \Sigma_\times$, let $c(x, y)$ denote the number of common features between $x$ and $y$. The next notation follows that introduced in [7]. Thus, let $s, t$ be two sequences over the finite set $\Sigma_\times$, and let $|s|$ denote the length of $s = s_1...s_{|s|}$. The sequence $s[i:j]$ is the contiguous subsequence $s_i...s_j$ of $s$. Let $\mathbf{i} = (i_1, ..., i_{|\mathbf{i}|})$ be a sequence of $|\mathbf{i}|$ indices in $s$, in ascending order. We define the length $l(\mathbf{i})$ *of the index sequence* $\mathbf{i}$ *in* $s$ as $i_{|\mathbf{i}|} - i_1 + 1$. Similarly, $\mathbf{j}$ is a sequence of $|\mathbf{j}|$ indices in $t$.

Let $\Sigma_\cup = \Sigma_1 \cup \Sigma_2 \cup ... \cup \Sigma_k$ be the set of all possible features. We say that the sequence $u \in \Sigma_\cup^*$ is a (sparse) subsequence of $s$ if there is a sequence of $|u|$ indices $\mathbf{i}$ such that $u_k \in s_{i_k}$, for all $k = 1, ..., |u|$. Equivalently, we write $u \prec s[\mathbf{i}]$ as a shorthand for the component-wise '$\in$' relationship between $u$ and $s[\mathbf{i}]$.

Finally, let $K_n(s, t, \lambda)$ (Equation 1) be the number of weighted sparse subsequences $u$ of length $n$ common to $s$ and $t$ (i.e. $u \prec s[\mathbf{i}]$, $u \prec t[\mathbf{j}]$), where the weight of $u$ is $\lambda^{l(\mathbf{i})+l(\mathbf{j})}$, for some $\lambda \leq 1$.

$$K_n(s, t, \lambda) = \sum_{u \in \Sigma_\cup^n} \sum_{\mathbf{i}:u \prec s[\mathbf{i}]} \sum_{\mathbf{j}:u \prec t[\mathbf{j}]} \lambda^{l(\mathbf{i})+l(\mathbf{j})} \tag{1}$$

Because for two fixed index sequences $\mathbf{i}$ and $\mathbf{j}$, both of length $n$, the size of the set $\{u \in \Sigma_\cup^n | u \prec s[\mathbf{i}], u \prec t[\mathbf{j}]\}$ is $\prod_{k=1}^n c(s_{i_k}, t_{j_k})$, then we can rewrite $K_n(s, t, \lambda)$ as in Equation 2:

$$K_n(s, t, \lambda) = \sum_{\mathbf{i}:|\mathbf{i}|=n} \sum_{\mathbf{j}:|\mathbf{j}|=n} \prod_{k=1}^n c(s_{i_k}, t_{j_k}) \lambda^{l(\mathbf{i})+l(\mathbf{j})} \tag{2}$$

We use $\lambda$ as a decaying factor that penalizes longer subsequences. For sparse subsequences, this means that wider gaps will be penalized more, which is exactly the desired behavior for our patterns. Through them, we try to capture head-modifier dependencies that are important for relation extraction; for lack of reliable dependency information, the larger the word gap is between two words, the less confident we are in the existence of a head-modifier relationship between them.

To enable an efficient computation of $K_n$, we use the auxiliary function $K_n'$ with a similar definition as $K_n$, the only difference being that it counts the length from the beginning of the particular subsequence $u$ to the end of the strings $s$ and $t$, as illustrated in Equation 3:

$$K_n'(s, t, \lambda) = \sum_{u \in \Sigma_\cup^n} \sum_{\mathbf{i}: u \prec s[\mathbf{i}]} \sum_{\mathbf{j}: u \prec t[\mathbf{j}]} \lambda^{|s| + |t| - i_1 - j_1 + 2} \tag{3}$$

An equivalent formula for $K_n'(s, t, \lambda)$ is obtained by changing the exponent of $\lambda$ from Equation 2 to $|s| + |t| - i_1 - j_1 + 2$.

Based on all definitions above, $K_n$ can be computed in $O(kn|s||t|)$ time, by modifying the recursive computation from [7] with the new factor $c(x, y)$, as shown in Figure 1. In this figure, the sequence $sx$ is the result of appending $x$ to $s$ (with $ty$ defined in a similar way). To avoid clutter, the parameter $\lambda$ is not shown in the argument list of $K$ and $K'$, unless it is instantiated to a specific constant.

$$
\begin{aligned}
K_0'(s, t) &= 1, \, for \, all \, s, t \\
K_i''(sx, ty) &= \lambda K_i''(sx, t) + \lambda^2 K_{i-1}'(s, t) \cdot c(x, y) \\
K_i'(sx, t) &= \lambda K_i'(s, t) + K_i''(sx, t) \\
K_n(sx, t) &= K_n(s, t) + \sum_j \lambda^2 K_{n-1}'(s, t[1:j-1]) \cdot c(x, t[j])
\end{aligned}
$$

Figure 1: Computation of subsequence kernel.

## 4.2 Computing the Relation Kernel

As described in Section 2, the input consists of a set of sentences, where each sentence contains exactly two entities (protein names in the case of interaction extraction). In Figure 2 we show the segments that will be used for computing the relation kernel between two example sentences $s$ and $t$. In sentence $s$ for instance, $x_1$ and $x_2$ are the two entities, $s_f$ is the sentence segment before $x_1$, $s_b$ is the segment between $x_1$ and $x_2$, and $s_a$ is the sentence segment after $x_2$. For convenience, we also include the auxiliary segment $s_b' = x_1 s_b x_2$, whose span is computed as $l(s_b') = l(s_b) + 2$ (in all length computations, we consider $x_1$ and $x_2$ as contributing one unit only).

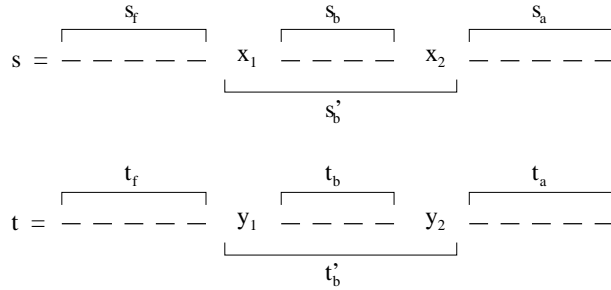

Figure 2: Sentence segments.

The relation kernel computes the number of common patterns between two sentences $s$ and $t$, where the set of patterns is restricted to the three types introduced in Section 3. Therefore, the kernel $rK(s, t)$ is expressed as the sum of three sub-kernels: $fbK(s, t)$ counting the

$$
\begin{aligned}
rK(s,t) &= fbK(s,t) + bK(s,t) + baK(s,t) \\
bK_i(s,t) &= K_i(s_b, t_b, 1) \cdot c(x_1, y_1) \cdot c(x_2, y_2) \cdot \lambda^{l(s_b') + l(t_b')} \\
fbK(s,t) &= \sum_{i,j} bK_i(s,t) \cdot K_j'(s_f, t_f), \quad 1 \le \text{i}, \ 1 \le \text{j}, \ \text{i}+\text{j} < \text{fb}_{\max} \\
bK(s,t) &= \sum_{i} bK_i(s,t), \quad 1 \le \text{i} \le \text{b}_{\max} \\
baK(s,t) &= \sum_{i,j} bK_i(s,t) \cdot K_j'(s_a^-, t_a^-), \quad 1 \le \text{i}, \ 1 \le \text{j}, \ \text{i}+\text{j} < \text{ba}_{\max}
\end{aligned}
$$

Figure 3: Computation of relation kernel.

number of common fore–between patterns, $bK(s,t)$ for between patterns, and $baK(s,t)$ for between–after patterns, as in Figure 3.

All three sub-kernels include in their computation the counting of common subsequences between $s_b'$ and $t_b'$. In order to speed up the computation, all these common counts can be calculated separately in $bK_i$, which is defined as the number of common subsequences of length $i$ between $s_b'$ and $t_b'$, anchored at $x_1/x_2$ and $y_1/y_2$ respectively (i.e. constrained to start at $x_1/y_1$ and to end at $x_2/y_2$). Then $fbK$ simply counts the number of subsequences that match $j$ positions before the first entity and $i$ positions between the entities, constrained to have length less than a constant $fb_{max}$. To obtain a similar formula for $baK$ we simply use the reversed (mirror) version of segments $s_a$ and $t_a$ (e.g. $s_a^-$ and $t_a^-$). In Section 3 we observed that all three subsequence patterns use at most 4 words to express a relation, therefore we set constants $fb_{max}$, $b_{max}$ and $ba_{max}$ to 4. Kernels $K$ and $K'$ are computed using the procedure described in Section 4.1.

## 5 Experimental Results

The relation kernel (ERK) is evaluated on the task of extracting relations from two corpora with different types of narrative, which are described in more detail in the following sections. In both cases, we assume that the entities and their labels are known. All pre-processing steps – sentence segmentation, tokenization, POS tagging and chunking – were performed using the OpenNLP[1] package. If a sentence contains $n$ entities ($n \ge 2$), it is replicated into $\binom{n}{2}$ sentences, each containing only two entities. If the two entities are known to be in a relationship, then the replicated sentence is added to the set of corresponding positive sentences, otherwise it is added to the set of negative sentences. During testing, a sentence having $n$ entities ($n \ge 2$) is again replicated into $\binom{n}{2}$ sentences in a similar way.

The relation kernel is used in conjunction with SVM learning in order to find a decision hyperplane that best separates the positive examples from negative examples. We modified the LibSVM[2] package by plugging in the kernel described above. In all experiments, the decay factor $\lambda$ is set to $0.75$. The performance is measured using *precision* (percentage of correctly extracted relations out of total extracted) and *recall* (percentage of correctly extracted relations out of total number of relations annotated in the corpus). When PR curves are reported, the precision and recall are computed using output from 10-fold cross-validation. The graph points are obtained by varying a threshold on the minimum acceptable extraction confidence, based on the probability estimates from LibSVM.

## 5.1 Interaction Extraction from AImed

We did comparative experiments on the AImed corpus, which has been previously used for training the protein interaction extraction systems in [6]. It consists of 225 Medline abstracts, of which 200 are known to describe interactions between human proteins, while the other 25 do not refer to any interaction. There are 4084 protein references and around 1000 tagged interactions in this dataset.

We compare the following three systems on the task of retrieving protein interactions from AImed (assuming gold standard proteins):

• **[Manual]**: We report the performance of the rule-based system of [3, 4].

• **[ELCS]**: We report the 10-fold cross-validated results from [6] as a PR graph.

• **[ERK]**: Based on the same splits as ELCS, we compute the corresponding precision-recall graph. In order to have a fair comparison with the other two systems, which use only lexical information, we do not use any word classes here.

The results, summarized in Figure 4(a), show that the relation kernel outperforms both ELCS and the manually written rules.

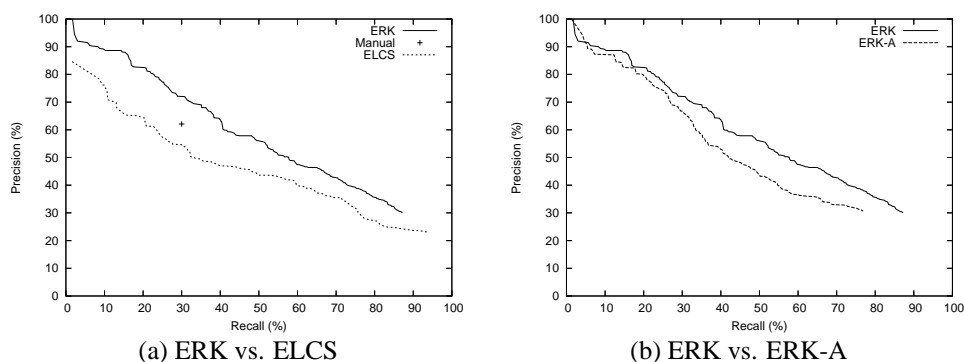

(a) ERK vs. ELCS          (b) ERK vs. ERK-A

Figure 4: PR curves for interaction extractors.

To evaluate the impact that the three types of patterns have on performance, we compare ERK with an ablated system (ERK-A) that uses all possible patterns, constrained only to be anchored on the two entity names. As can be seen in Figure 4(b), the three patterns (FB, B, BA) do lead to a significant increase in performance, especially for higher recall levels.

## 5.2 Relation Extraction from ACE

To evaluate how well this relation kernel ports to other types of narrative, we applied it to the problem of extracting top-level relations from the ACE corpus [2], the version used for the September 2002 evaluation. The training part of this dataset consists of 422 documents, with a separate set of 97 documents allocated for testing. This version of the ACE corpus contains three types of annotations: coreference, named entities and relations. There are five types of entities – PERSON, ORGANIZATION, FACILITY, LOCATION, and GEO-POLITICAL ENTITY – which can participate in five general, top-level relations: ROLE, PART, LOCATED, NEAR, and SOCIAL. A recent approach to extracting relations is described in [9]. The authors use a generalized version of the tree kernel from [10] to compute a kernel over relation examples, where a relation example consists of the smallest dependency tree containing the two entities of the relation. Precision and recall values are reported for the task of extracting the 5 top-level relations in the ACE corpus under two different scenarios:

– **[S1]** This is the classic setting: one multi-class SVM is learned to discriminate among

the 5 top-level classes, plus one more class for the no-relation cases.

– **[S2]** One binary SVM is trained for *relation detection*, meaning that all positive relation instances are combined into one class. The thresholded output of this binary classifier is used as training data for a second multi-class SVM, trained for *relation classification*.

We trained our relation kernel, under the first scenario, to recognize the same 5 top-level relation types. While for interaction extraction we used only the lexicalized version of the kernel, here we utilize more features, corresponding to the following feature spaces: $\Sigma_1$ is the word vocabulary, $\Sigma_2$ is the set of POS tags, $\Sigma_3$ is the set of generic POS tags, and $\Sigma_4$ contains the 5 entity types. We also used chunking information as follows: all (sparse) subsequences were created exclusively from the chunk heads, where a head is defined as the last word in a chunk. The same criterion was used for computing the length of a subsequence – all words other than head words were ignored. This is based on the observation that in general words other than the chunk head do not contribute to establishing a relationship between two entities outside of that chunk. One exception is when both entities in the example sentence are contained in the same chunk. This happens very often due to noun-noun ('U.S. troops') or adjective-noun ('Serbian general') compounds. In these cases, we let one chunk contribute both entity heads. Also, an important difference from the interaction extraction case is that often the two entities in a relation do not have any words separating them, as for example in noun-noun compounds. None of the three patterns from Section 3 capture this type of dependency, therefore we introduced a fourth type of pattern, the modifier pattern **M**. This pattern consists of a sequence of length two formed from the head words (or their word classes) of the two entities. Correspondingly, we updated the relation kernel from Figure 3 with a new kernel term $mK$, as illustrated in Equation 4.

$$rK(s,t) = fbK(s,t) + bK(s,t) + baK(s,t) + mK(s,t) \qquad (4)$$

The sub-kernel $mK$ corresponds to a product of counts, as shown in Equation 5.

$$mK(s,t) = c(x_1, y_1) \cdot c(x_2, y_2) \cdot \lambda^{2+2} \qquad (5)$$

We present in Table 1 the results of using our updated relation kernel to extract relations from ACE, under the first scenario. We also show the results presented in [9] for their best performing kernel K4 (a sum between a bag-of-words kernel and the dependency kernel) under both scenarios.

Table 1: Extraction Performance on ACE.

| Method | Precision | Recall | F-measure |
|---|---|---|---|
| (S1) ERK | **73.9** | **35.2** | **47.7** |
| (S1) K4 | 70.3 | 26.3 | 38.0 |
| (S2) K4 | 67.1 | 35.0 | 45.8 |

Even though it uses less sophisticated syntactic and semantic information, ERK in S1 significantly outperforms the dependency kernel. Also, ERK already performs a few percentage points better than K4 in S2. Therefore we expect to get an even more significant increase in performance by training our relation kernel in the same cascaded fashion.

## 6 Related Work

In [10], a tree kernel is defined over shallow parse representations of text, together with an efficient algorithm for computing it. Experiments on extracting PERSON-AFFILIATION and ORGANIZATION-LOCATION relations from 200 news articles show the advantage of using this new type of tree kernels over three feature-based algorithms. The same kernel was slightly generalized in [9] and applied on dependency tree representations of sentences, with dependency trees being created from head-modifier relationships extracted from syntactic parse trees. Experimental results show a clear win of the dependency tree kernel over a bag-of-words kernel. However, in a bag-of-words approach the word order is completely lost. For relation extraction, word order is important, and our experimental results support this claim – all subsequence patterns used in our approach retain the order between words.

The tree kernels used in the two methods above are *opaque* in the sense that the semantics of the dimensions in the corresponding Hilbert space is not obvious. For subsequence kernels, the semantics is known by definition: each subsequence pattern corresponds to a dimension in the Hilbert space. This enabled us to easily restrict the types of patterns counted by the kernel to the three types that we deemed relevant for relation extraction.

# 7  Conclusion and Future Work

We have presented a new relation extraction method based on a generalization of subsequence kernels. When evaluated on a protein interaction dataset, the new method showed better performance than two previous rule-based systems. After a small modification, the same kernel was evaluated on the task of extracting top-level relations from the ACE corpus, showing better performance when compared with a recent dependency tree kernel.

An experiment that we expect to lead to better performance was already suggested in Section 5.2 – using the relation kernel in a cascaded fashion, in order to improve the low recall caused by the highly unbalanced data distribution. Another performance gain may come from setting the factor $\lambda$ to a more appropriate value based on a development dataset.

Currently, the method assumes the named entities are known. A natural extension is to integrate named entity recognition with relation extraction. Recent research [11] indicates that a global model that captures the mutual influences between the two tasks can lead to significant improvements in accuracy.

# 8  Acknowledgements

This work was supported by grants IIS-0117308 and IIS-0325116 from the NSF. We would like to thank Rohit J. Kate and the anonymous reviewers for helpful observations.

## Footnotes

[1]URL: http://opennlp.sourceforge.net

[2]URL:http://www.csie.ntu.edu.tw/~cjlin/libsvm/

# References

[1] R. Grishman, Message Understanding Conference 6, http://cs.nyu.edu/cs/faculty/grishman/muc6.html (1995).

[2] NIST, ACE – Automatic Content Extraction, http://www.nist.gov/speech/tests/ace (2000).

[3] C. Blaschke, A. Valencia, Can bibliographic pointers for known biological data be found automatically? protein interactions as a case study, Comparative and Functional Genomics 2 (2001) 196–206.

[4] C. Blaschke, A. Valencia, The frame-based module of the Suiseki information extraction system, IEEE Intelligent Systems 17 (2002) 14–20.

[5] S. Ray, M. Craven, Representing sentence structure in hidden Markov models for information extraction, in: Proceedings of the Seventeenth International Joint Conference on Artificial Intelligence (IJCAI-2001), Seattle, WA, 2001, pp. 1273–1279.

[6] R. Bunescu, R. Ge, R. J. Kate, E. M. Marcotte, R. J. Mooney, A. K. Ramani, Y. W. Wong, Comparative experiments on learning information extractors for proteins and their interactions, Artificial Intelligence in Medicine (special issue on Summarization and Information Extraction from Medical Documents) 33 (2) (2005) 139–155.

[7] H. Lodhi, C. Saunders, J. Shawe-Taylor, N. Cristianini, C. Watkins, Text classification using string kernels, Journal of Machine Learning Research 2 (2002) 419–444.

[8] V. N. Vapnik, Statistical Learning Theory, John Wiley & Sons, 1998.

[9] A. Culotta, J. Sorensen, Dependency tree kernels for relation extraction, in: Proceedings of the 42nd Annual Meeting of the Association for Computational Linguistics (ACL-04), Barcelona, Spain, 2004, pp. 423–429.

[10] D. Zelenko, C. Aone, A. Richardella, Kernel methods for relation extraction, Journal of Machine Learning Research 3 (2003) 1083–1106.

[11] D. Roth, W. Yih, A linear programming formulation for global inference in natural language tasks, in: Proceedings of the Annual Conference on Computational Natural Language Learning (CoNLL), Boston, MA, 2004, pp. 1–8.
